# Feature Selection in Mixture-Based Clustering

**Martin H. Law, Anil K. Jain**
Dept. of Computer Science and Eng.
Michigan State University,
East Lansing, MI 48824
**U.S.A.**

**Mário A. T. Figueiredo**
Instituto de Telecomunicações,
Instituto Superior Técnico
1049-001 Lisboa
**Portugal**

## Abstract

There exist many approaches to clustering, but the important issue of feature selection, *i.e.*, selecting the data attributes that are relevant for clustering, is rarely addressed. Feature selection for clustering is difficult due to the absence of class labels. We propose two approaches to feature selection in the context of Gaussian mixture-based clustering. In the first one, instead of making hard selections, we estimate feature saliencies. An expectation-maximization (EM) algorithm is derived for this task. The second approach extends Koller and Sahami's mutual-information-based feature relevance criterion to the unsupervised case. Feature selection is then carried out by a backward search scheme. This scheme can be classified as a "wrapper", since it wraps mixture estimation in an outer layer that performs feature selection. Experimental results on synthetic and real data show that both methods have promising performance.

## 1   Introduction

In partitional clustering, each pattern is represented by a vector of *features*. However, not all the features are useful in constructing the partitions: some features may be just noise, thus not contributing to (or even degrading) the clustering process. The task of selecting the "best" feature subset, known as *feature selection* (FS), is therefore an important task. In addition, FS may lead to more economical clustering algorithms (both in storage and computation) and, in many cases, it may contribute to the interpretability of the models. FS is particularly relevant for data sets with large numbers of features; *e.g.*, on the order of thousands as seen in some molecular biology [22] and text clustering applications [21].

In supervised learning, FS has been widely studied, with most methods falling into two classes: *filters*, which work independently of the subsequent learning algorithm; *wrappers*, which use the learning algorithm to evaluate feature subsets [12]. In contrast, FS has received little attention in clustering, mainly because, without class labels, it is unclear how to assess feature relevance. The problem is even more difficult when the number of clusters is unknown, since the number of clusters and the best feature subset are inter-related [6].

Some approaches to FS in clustering have been proposed. Of course, any method not

---

Email addresses: lawhiu@cse.msu.edu, jain@cse.msu.edu, mtf@lx.it.pt
This work was supported by the U.S. Office of Naval Research, grant no. 00014-01-1-0266, and by the Portuguese Foundation for Science and Technology, project POSI/33143/SRI/2000.

relying on class labels (*e.g.*, [16]) can be used. Dy and Brodley [6] suggested a heuristic to compare feature subsets, using cluster separability. A Bayesian approach for multinomial mixtures was proposed in [21]; another Bayesian approach using a shrinkage prior was considered in [8]. Dash and Liu [4] assess the clustering tendency of each feature by an entropy index. A genetic algorithm was used in [11] for FS in $k$-means clustering. Talavera [19] addressed FS for symbolic data. Finally, Devaney and Ram [5] use a notion of "category utility" for FS in conceptual clustering, and Modha and Scott-Spangler [17] assign weights to feature groups with a score similar to Fisher discrimination.

In this paper, we introduce two new FS approaches for mixture-based clustering [10, 15]. The first is based on a *feature saliency* measure which is obtained by an EM algorithm; unlike most FS methods, this does not involve any explicit search. The second approach extends the mutual-information based criterion of [13] to the unsupervised context; it is a *wrapper*, since FS is *wrapped around* a basic mixture estimation algorithm.

## 2 Finite Mixtures and the EM algorithm

Given $N$ i.i.d. samples $\mathcal{Y} = \{\mathbf{y}_1, ..., \mathbf{y}_N\}$, the log-likelihood of a $K$-component mixture is

$$\log p(\mathcal{Y}|\boldsymbol{\theta}) \; = \; \log \prod_{i=1}^{N} p(\mathbf{y}_i|\boldsymbol{\theta}) = \sum_{i=1}^{N} \log \sum_{j=1}^{K} \alpha_j \, p(\mathbf{y}_i|\boldsymbol{\theta}_j), \tag{1}$$

where: $\forall_j, \alpha_j \geq 0$; $\sum_j \alpha_j = 1$; each $\boldsymbol{\theta}_j$ is the set of parameters of the $j$-th component; and $\boldsymbol{\theta} \equiv \{\boldsymbol{\theta}_1, ..., \boldsymbol{\theta}_K, \alpha_1, ..., \alpha_K\}$ is the full parameter set. Each $\mathbf{y}_i$ is a $D$-dimensional feature vector $[y_{i,1}, ..., y_{i,D}]^T$ and all components have the same form (*e.g.*, Gaussian).

Neither *maximum likelihood* ($\widehat{\boldsymbol{\theta}}_{\text{ML}} = \arg\max_{\boldsymbol{\theta}} \{\log p(\mathcal{Y}|\boldsymbol{\theta})\}$) nor *maximum a posteriori* ($\widehat{\boldsymbol{\theta}}_{\text{MAP}} = \arg\max_{\boldsymbol{\theta}} \{\log p(\mathcal{Y}|\boldsymbol{\theta}) + \log p(\boldsymbol{\theta})\}$) estimates can be found analytically. The usual choice is the EM algorithm, which finds local maxima of these criteria. Let $\mathcal{Z} = \{\mathbf{z}_1, ..., \mathbf{z}_N\}$ be a set of $N$ *missing* labels, where $\mathbf{z}_i = [z_{i,1}, ..., z_{i,K}]$, with $z_{i,j} = 1$ and $z_{i,p} = 0$, for $p \neq j$, meaning that $\mathbf{y}_i$ is a sample of $p(\cdot|\boldsymbol{\theta}_j)$. The complete log-likelihood is

$$\log p(\mathcal{Y}, \mathcal{Z}|\boldsymbol{\theta}) = \sum_{i=1}^{N} \sum_{j=1}^{K} z_{i,j} \log \left[\alpha_j p(\mathbf{y}_i|\boldsymbol{\theta}_j)\right]. \tag{2}$$

EM produces a sequence of estimates $\{\widehat{\boldsymbol{\theta}}(t), \ t = 0, 1, 2, ...\}$ using two alternating steps:

• **E-step:** Computes $\mathcal{W} = E[\mathcal{Z}|\mathcal{Y}, \widehat{\boldsymbol{\theta}}(t)]$, and plugs it into $\log p(\mathcal{Y}, \mathcal{Z}|\boldsymbol{\theta})$ yielding the $Q$-function $Q(\boldsymbol{\theta}, \widehat{\boldsymbol{\theta}}(t)) = \log p(\mathcal{Y}, \mathcal{W}|\boldsymbol{\theta})$. Since the elements of $\mathcal{Z}$ are binary, we have

$$w_{i,j} \equiv E\left[z_{i,j}|\mathcal{Y}, \widehat{\boldsymbol{\theta}}(t)\right] = \Pr\left[z_{i,j} = 1|\mathbf{y}_i, \widehat{\boldsymbol{\theta}}(t)\right] \propto \widehat{\alpha}_j(t) \, p(\mathbf{y}_i|\widehat{\boldsymbol{\theta}}_j(t)), \tag{3}$$

followed by normalization so that $\sum_j w_{i,j} = 1$. Notice that $\alpha_j$ is the *a priori* probability that $z_{i,j} = 1$ (*i.e.*, $\mathbf{y}_i$ belongs to cluster $j$) while $w_{i,j}$ is the corresponding *a posteriori* probability, after observing $\mathbf{y}_i$.

• **M-step:** Updates the parameter estimates, $\widehat{\boldsymbol{\theta}}(t+1) = \arg\max_{\boldsymbol{\theta}} \{Q(\boldsymbol{\theta}, \widehat{\boldsymbol{\theta}}(t)) + \log p(\boldsymbol{\theta})\}$, in the case of MAP estimation, or without $\log p(\boldsymbol{\theta})$ in the ML case.

## 3 A Mixture Model with Feature Saliency

In our first approach to FS, we assume conditionally independent features, given the component label (which in the Gaussian case corresponds to diagonal covariance matrices),

$$p(\mathbf{y}|\{\alpha_j\}, \{\theta_{jl}\}) = \sum_{j=1}^{K} \alpha_j p(\mathbf{y}|\theta_j) = \sum_{j=1}^{K} \alpha_j \prod_{l=1}^{D} p(y_l|\theta_{jl}), \tag{4}$$

where $p(\cdot|\theta_{jl})$ is the pdf of the $l$-th feature in the $j$-th component; in general, this could have any form, although we only consider Gaussian densities. In the sequel, we will use the indices $i$, $j$ and $l$ to run through data points, mixture components, and features, respectively. Assume now that some features are *irrelevant*, in the following sense: if feature $l$ is irrelevant, then $p(y_l|\theta_{jl}) = q(y_l|\lambda_l)$, for $j = 1, ..., K$, where $q(y_l|\lambda_l)$ is the common (*i.e.*, independent of $j$) density of feature $l$. Let $\Phi = (\phi_1, ..., \phi_D)$ be a set of binary parameters, such that $\phi_l = 1$ if feature $l$ is relevant and $\phi_l = 0$ otherwise; then,

$$p(\mathbf{y}|\Phi, \{\alpha_j\}, \{\theta_{jl}\}, \{\lambda_l\}) = \sum_{j=1}^{K} \alpha_j \prod_{l=1}^{D} (p(y_l|\theta_{jl}))^{\phi_l} (q(y_l|\lambda_l))^{1-\phi_l}. \tag{5}$$

Our approach consists of: (i) treating the $\phi_l$'s as missing variables rather than as parameters; (ii) estimating $\rho_l = P(\phi_l = 1)$ from the data; $\rho_l$ is the probability that the $l$-th feature is useful, which we call its *saliency*. The resulting mixture model (see proof in [14]) is

$$p(\mathbf{y}|\{\alpha_j\}, \{\theta_{jl}\}, \{\lambda_l\}, \{\rho_l\}) = \sum_{j=1}^{K} \alpha_j \prod_{l=1}^{D} \big(\rho_l p(y_l|\theta_{jl}) + (1 - \rho_l)q(y_l|\lambda_l)\big). \tag{6}$$

The form of $q(.|.)$ reflects our prior knowledge about the distribution of the non-salient features. In principle, it can be any 1-D pdf (*e.g.*, Gaussian or student-t); here we only consider $q(.|.)$ to be a Gaussian. Equation (6) has a generative interpretation. As in a standard finite mixture, we first select the component label $j$ by sampling from a multinomial distribution with parameters $(\alpha_1, \ldots, \alpha_K)$. Then, for each feature $l = 1, ..., D$, we flip a biased coin whose probability of getting a head is $\rho_l$; if we get a head, we use the mixture component $p(.|\theta_{jl})$ to generate the $l$-th feature; otherwise, the common component $q(.|\lambda_l)$ is used.

Given a set of observations $\mathcal{Y} = (\mathbf{y}_1, \ldots, \mathbf{y}_N)$, with $\mathbf{y}_i = [y_{i,1}, ..., y_{i,D}]^T$, the parameters $\Theta = (\{\alpha_j\}, \{\theta_{jl}\}, \{\lambda_l\}, \{\rho_l\})$ can be estimated by the maximum likelihood criterion,

$$\widehat{\Theta} = \arg\max_{\Theta} \sum_{i=1}^{N} \log \sum_{j=1}^{K} \alpha_j \prod_{l=1}^{D} \big(\rho_l p(y_{il}|\theta_{jl}) + (1 - \rho_l)q(y_{il}|\lambda_l)\big). \tag{7}$$

In the absence of a closed-form solution, an EM algorithm can be derived by treating both the $z_i$'s and the $\phi_l$'s as missing data (see [14] for details).

### 3.1 Model Selection

Standard EM for mixtures exhibits some weaknesses which also affect the EM algorithm just mentioned: it requires knowledge of $K$, and a good initialization is essential for reaching a good local optimum. To overcome these difficulties, we adopt the approach in [9], which is based on the MML criterion [23, 24]. The MML criterion for the proposed model (see details in [14]) consists of minimizing, with respect to $\Theta$, the following cost function

$$-\log p(\mathcal{Y}|\Theta) + \frac{K+D}{2} \log N + \frac{R}{2} \sum_{l=1}^{D} \sum_{j=1}^{K} \log(N\alpha_j\rho_l) + \frac{S}{2} \sum_{l=1}^{D} \log(N(1 - \rho_l)), \tag{8}$$

where $R$ and $S$ are the number of parameters in $\theta_{jl}$ and $\lambda_l$, respectively. If $p(.|.)$ and $q(.|.)$ are univariate Gaussians (arbitrary mean and variance), $R = S = 2$. From a parameter estimation viewpoint, this is equivalent to a MAP estimate with conjugate (improper) Dirichlet-type priors on the $\alpha_j$'s and $\rho_l$'s (see details in [14]); thus, the EM algorithm undergoes a minor modification in the M-step, which still has a closed form.

The terms in equation (8), in addition to the log-likelihood $-\log p(\mathcal{Y}|\Theta)$, have simple interpretations. The term $\frac{K+D}{2} \log N$ is a standard MDL-type parameter code-length corresponding to $K$ $\alpha_j$ values and $D$ $\rho_l$ values. For the $l$-th feature in the $j$-th component, the

"effective" number of data points for estimating $\theta_{jl}$ is $N\alpha_j\rho_l$. Since there are $R$ parameters in each $\theta_{jl}$, the corresponding code-length is $\frac{R}{2}\log(n\rho_l\alpha_j)$. Similarly, for the $l$-th feature in the common component, the number of effective data points for estimation is $N(1-\rho_l)$. Thus, there is a term $\frac{S}{2}\log(N(1-\rho_l))$ in (8) for each feature.

One key property of the EM algorithm for minimizing equation (8) is its pruning behavior, forcing some of the $\alpha_j$ to go to zero and some of the $\rho_l$ to go to zero or one. Worries that the message length in (8) may become invalid at these boundary values can be circumvented by the arguments in [9]. When $\rho_l$ goes to zero, the $l$-th feature is no longer salient and $\rho_l$ and $\theta_{1l}, \ldots, \theta_{Kl}$ are removed. When $\rho_l$ goes to 1, $\lambda_l$ and $\rho_l$ are dropped.

Finally, since the model selection algorithm determines the number of components, it can be initialized with a large value of $K$, thus alleviating the need for a good initialization [9]. Because of this, as in [9], a component-wise version of EM [2] is adopted (see [14]).

## 3.2 Experiments and Results

The first data set considered consists of 800 points from a mixture of 4 equiprobable Gaussians with mean vectors $\begin{pmatrix} 0 \\ 3 \end{pmatrix}$, $\begin{pmatrix} 1 \\ 9 \end{pmatrix}$, $\begin{pmatrix} 6 \\ 4 \end{pmatrix}$, $\begin{pmatrix} 7 \\ 10 \end{pmatrix}$, and identity covariance matrices. Eight "noisy" features (sampled from a $\mathcal{N}(0,1)$ density) were appended to this data, yielding a set of 800 10-D patterns. The proposed algorithm was run 10 times, each initialized with $K = 30$; the common component is initialized to cover all data, and the feature saliencies are initialized at 0.5. In all the 10 runs, the 4 components were always identified. The saliencies of all the ten features, together with their standard deviations (error bars), are shown in Fig. 1. We conclude that, in this case, the algorithm successfully locates the clusters and correctly assigns the feature saliencies. See [14] for more details on this experiment.

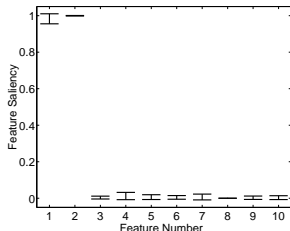

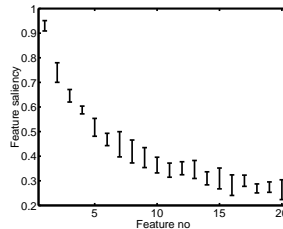

Figure 1: Feature saliency for 10-D 4-component Gaussian mixture. Only the first two features are relevant. The error bars show ± one standard deviation.

Figure 2: Feature saliency for the Trunk data. The smaller the feature number, the more important is the feature.

In the next experiment, we consider Trunk's data [20], which has two 20-dimensional Gaussians classes with means $\mathbf{m}_1 = (1, \frac{1}{\sqrt{2}}, \ldots, \frac{1}{\sqrt{20}})$ and $\mathbf{m}_2 = -\mathbf{m}_1$, and covariances $\mathbf{C}_1 = \mathbf{C}_2 = \mathbf{I}$. Data is obtained by sampling 5000 points from each of these two Gaussians. Note that these features have a descending order of relevance. As above, the initial $K$ is set to 30. In all the 10 runs performed, two components were always detected. The values of the feature saliencies are shown in Fig. 2. We see the general trend that as the feature number increases, the saliency decreases, following the true characteristics of the data.

Feature saliency values were also computed for the "wine" data set (available at the UCI repository at www.ics.uci.edu/~mlearn/MLRepository.html), consisting of 178 13-dimensional points in three classes. After standardizing all features to zero mean and unit variance, we applied the *LNKnet* supervised feature selection algorithm (available at www.ll.mit.edu/IST/lnknet/). The nine features selected by *LNKnet* are 7, 13, 1, 5, 10, 2, 12, 6, 9. Our feature saliency algorithm (with no class labels) yielded the values

Table 1: Feature saliency of wine data

| 1 | 2 | 3 | 4 | 5 | 6 | 7 | 8 | 9 | 10 | 11 | 12 | 13 |
|------|------|------|------|------|------|------|------|------|------|------|------|------|
| 0.94 | 0.77 | 0.10 | 0.59 | 0.14 | 0.99 | 1.00 | 0.66 | 0.94 | 0.85 | 0.88 | 1.00 | 0.83 |

in Table 1. Ranking the features in descending order of saliency, we get the ordering: 7, 12, 6, 1, 9, 11, 10, 13, 2, 8, 4, 5, 3. The top 5 features (7, 12, 6, 1, 9) are all in the subset selected by *LNKnet*. If we skip the sixth feature (11), the following three features (10, 13, 2) were also selected by *LNKnet*. Thus we can see that for this data set, our algorithm, though totally unsupervised, performs comparably with a supervised feature selection algorithm.

## 4 A Feature Selection Wrapper

Our second approach is more traditional in the sense that it selects a feature subset, instead of estimating feature saliency. The number of mixture components is assumed known a priori, though no restriction on the covariance of the Gaussian components is imposed.

### 4.1 Irrelevant Features and Conditional Independence

Assume that the class labels, $\mathbf{z}$, and the full feature vector, $\mathbf{y}$, follow some joint probability function $p(\mathbf{z}, \mathbf{y})$. In supervised learning [13], a feature subset $\mathbf{y}_{\mathcal{N}}$ is considered irrelevant if it is conditionally independent of the label $\mathbf{z}$, given the remaining features $\mathbf{y}_{\mathcal{U}}$, that is, if $p(\mathbf{z}|\mathbf{y}) = p(\mathbf{z}|\mathbf{y}_{\mathcal{U}}, \mathbf{y}_{\mathcal{N}}) = p(\mathbf{z}|\mathbf{y}_{\mathcal{U}})$, where $\mathbf{y}$ is split into two subsets: "useful" features $\mathbf{y}_{\mathcal{U}}$ and "non-useful" features $\mathbf{y}_{\mathcal{N}}$ (here, $\mathcal{N} \subset \{1, ..., D\}$ is the index set of the non-useful features). It is easy to show that this implies

$$p(\mathbf{y}|\mathbf{z}) = p(\mathbf{y}_{\mathcal{U}}|\mathbf{z})p(\mathbf{y}_{\mathcal{N}}|\mathbf{z}, \mathbf{y}_{\mathcal{U}}) = p(\mathbf{y}_{\mathcal{U}}|\mathbf{z})p(\mathbf{y}_{\mathcal{N}}|\mathbf{y}_{\mathcal{U}}). \tag{9}$$

To generalize this notion to unsupervised learning, we propose to let the expectations $w_j$ (a byproduct of the EM algorithm) play the role of the missing class labels. Recall that the $w_j$ (see (3)) are posterior class probabilities, Prob$[\mathbf{y} \in \text{class } j | \mathbf{y}, \boldsymbol{\theta}]$. Consider the posterior probabilities based on all the features, and only on the useful features, respectively

$$w_{i,j} \propto \widehat{\alpha}_j \, p(\mathbf{y}_i | \widehat{\boldsymbol{\theta}}_j), \qquad v_{i,j}(\mathcal{N}) \propto \widehat{\alpha}_j \, p(\mathbf{y}_{i,\mathcal{U}} | \widehat{\boldsymbol{\theta}}_{j,\mathcal{U}}), \tag{10}$$

where $\mathbf{y}_{i,\mathcal{U}}$ is the subset of relevant features of sample $\mathbf{y}_i$ (of course, the $v_{i,j}$ and $w_{i,j}$ have to be normalized such that $\sum_j v_{i,j} = 1$ and $\sum_j w_{i,j} = 1$). If $\mathbf{y}_{\mathcal{N}}$ is a completely irrelevant feature subset, then $v_{i,j}$ equals $w_{i,j}$ exactly, because of the conditional independence in (9), applied to (3). In practice, such features rarely exist, though they do exhibit different degrees of irrelevance. So we follow the suggestion in [13], and find $\mathcal{N}$ that gives $v_{i,m}(\mathcal{N})$ as close to $w_{i,m}$ as possible. As both $w_{i,j}$ and $v_{i,j}(\mathcal{N})$ are probabilities, a natural criterion for assessing their closeness is the expected value of the *Kullback-Leibler divergence* (KLD, [3]). This criterion is computed as a sample mean

$$\Upsilon(\mathcal{N}) = \sum_{i=1}^{N} \sum_{j=1}^{K} w_{i,j} \log \frac{w_{i,j}}{v_{i,j}(\mathcal{N})} \tag{11}$$

in our case. A low value of $\Upsilon(\mathcal{N})$ indicates that the features in $\mathcal{N}$ are "almost" conditionally independent from the expected class labels, given the features in $\mathcal{U}$.

In practice, we start by obtaining reasonable initial estimates of $\{w_{i,j}\}$ by running EM using all the features, and set $\mathcal{N} = \{\}$. At each stage, we find the feature $q \notin \mathcal{N}$ such that $\Upsilon(\mathcal{N} \cup \{q\})$ is smallest and add it to $\mathcal{N}$. EM is then run again, using the features not in $\mathcal{N}$, to update the posterior probabilities $\{w_{i,j}\}$. The process is then repeated until only one feature remains, in what can be considered as a backward search algorithm that yields a sorting of the features by decreasing order of irrelevance.

## 4.2 The assignment entropy

Given a method to sort the features in the order of relevance, we now require a method to measure how good each subset is. Unlike in supervised learning, we can not resort to classification accuracy. We adopt the criterion that a clustering is good if the clusters are "crisp", *i.e.*, if, for every $i$, $w_{i,j} \simeq 1$ for some $j$. A natural way to formalize this is to consider the mean entropy of the $\{w_{i,j}\}$; that is, the clustering is considered to be good if $H(\{w_{i,j}\}) = -N^{-1} \sum_{i=1}^{N} \sum_{j=1}^{K} w_{i,j} \log w_{i,j}$ is small. In the sequel, we call $H$ "the entropy of the assignment". An important characteristic of the entropy is that it cannot increase when more features are used (because, for any random variables $X$, $Y$, and $Z$, $H(X|Y,Z) \leq H(X|Y)$, a fundamental inequality of information theory [3]; note that $H(\{w_{im}\})$ is a conditional entropy $H(\{w_{im}\}|\{\mathbf{y}_{i,\mathcal{U}}\})$). Moreover, $H(\{w_{im}\})$ exhibits a *diminishing returns* behavior (decreasing abruptly as the most relevant features are included, but changing little when less relevant features are used). Our empirical results show that $H$ indeed has a strong relationship with the quality of the clusters. Of course, during the backward search, one can also consider picking the next feature whose removal least increases $H$, rather than the one yielding the smallest KLD; both options are explored in the experiments. Finally, we mention that other minimum-entropy-type criteria have been recently used for clustering [7], [18], but not for feature selection.

## 4.3 Experiments

We have conducted experiments on data sets commonly used for supervised learning tasks. Since we are doing unsupervised learning, the class labels are, of course, withheld and only used for evaluation. The two heuristics for selecting the next feature to be removed (based on minimum KLD and minimum entropy) are considered in different runs. To assess clustering quality, we assign each data point to the Gaussian component that most likely generated it and then compare this labelling with the ground-truth. Table 2 summarizes the characteristics of the data sets for which results are reported here (all available from the UCI repository); we have also performed tests on other data sets achieving similar results.

The experimental results shown in Fig. 3 reveal that the general trend of the error rate agrees well with $H$. The error rates either have a minimum close to the "knee" of the H curve, or the curve becomes flat. The two heuristics for selecting the feature to be removed perform comparably. For the *cover type* data set, the DKL heuristic yields lower error rates than the one based on $H$, while the contrary happens for *image segmentation* and WBC datasets.

# 5  Concluding Remarks and Future Work

The two approaches for unsupervised feature selection herein proposed have different advantages and drawbacks. The first approach avoids explicit feature search and does not require a pre-specified number of clusters; however, it assumes that the features are conditionally independent, given the components. The second approach places no restriction on the covariances, but it does assume knowledge of the number of components. We believe that both approaches can be useful in different scenarios, depending on which set of assumptions fits the given data better.

Several issues require further work: weakly relevant features (in the sense of [12]) are not removed by the first algorithm while the second approach relies on a good initial clustering. Overcoming these problems will make the methods more generally applicable. We also need to investigate the scalability of the proposed algorithms; ideas such as those in [1] can be exploited.

Table 2: Some details of the data sets (*WBC* stands for *Wisconsin breast cancer*).

| Name | cover type | image segmentation | WBC | wine |
|---|---|---|---|---|
| No. points used | 2000 | 1000 | 569 | 178 |
| No. of features | 10 | 18 | 30 | 13 |
| No. of classes | 4 | 7 | 2 | 3 |

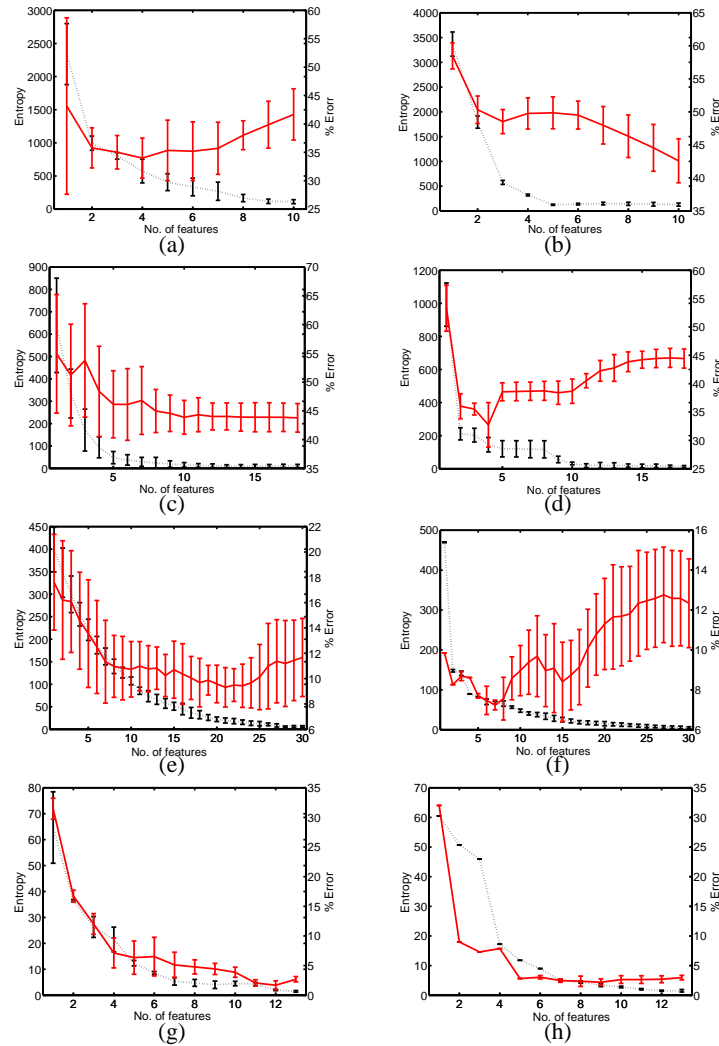

Figure 3: (a) and (b): *cover type*; (c) and (d): *image segmentation*; (e) and (f): WBC; (g) and (h): *wine*. Feature removal by minimum KLD (left column) and minimum $H$ (right column). Solid lines: error rates; dotted lines: $H$. Error bars correspond to $\pm$ one standard deviation over 10 runs.

# References

[1] P. Bradley, U. Fayyad, and C. Reina. Clustering very large database using EM mixture models. In *Proc. 15th Intern. Conf. on Pattern Recognition*, pp. 76–80, 2000.

[2] G. Celeux, S. Chrétien, F. Forbes, and A. Mkhadri. A component-wise EM algorithm for mixtures. *Journal of Computational and Graphical Statistics*, 10:699–712, 2001.

[3] T. Cover and J. Thomas. *Elements of Information Theory*. John Wiley & Sons, 1991.

[4] M. Dash and H. Liu. Feature selection for clustering. In *Proc. of Pacific-Asia Conference on Knowledge Discovery and Data Mining*, 2000, pp. 110–121.

[5] M. Devaney and A. Ram. Efficient feature selection in conceptual clustering. In *Proc. ICML'1997*, pp. 92–97, 1997.

[6] J. Dy and C. Brodley. Feature subset selection and order identification for unsupervised learning. In *Proc. ICML'2000*, pp. 247–254, 2000.

[7] E. Gokcay and J. Principe. Information Theoretic Clustering. *IEEE Trans. on PAMI*, 24(2):158-171, 2002.

[8] P. Gustafson, P. Carbonetto, N. Thompson, and N. de Freitas. Bayesian feature weighting for unsupervised learning, with application to object recognition. In *Proc. of the 9th Intern. Workshop on Artificial Intelligence and Statistics*, 2003.

[9] M. Figueiredo and A. Jain. Unsupervised learning of finite mixture models. *IEEE Trans. on PAMI*, 24(3):381–396, 2002.

[10] A. K. Jain and R. C. Dubes. *Algorithms for Clustering Data*. Prentice Hall, 1988.

[11] Y. Kim, W. Street, and F. Menczer. Feature Selection in Unsupervised Learning via Evolutionary Search. In *Proc. ACM SIGKDD*, pp. 365–369, 2000.

[12] R. Kohavi and G. John. Wrappers for feature subset selection. *Artificial Intelligence*, 97(1-2):273–324, 1997.

[13] D. Koller and M. Sahami. Toward optimal feature selection. In *Proc. ICML'1996*, pp. 284–292, 1996.

[14] M. Law, M. Figueiredo, and A. Jain. *Feature Saliency in Unsupervised Learning*. Tech. Rep., Dept. Computer Science and Eng., Michigan State Univ., 2002. Available at http://www.cse.msu.edu/~lawhiu/papers/TR02.ps.gz.

[15] G. McLachlan and K. Basford. *Mixture Models: Inference and Application to Clustering*. Marcel Dekker, New York, 1988.

[16] P. Mitra and C. A. Murthy. Unsupervised feature selection using feature similarity. *IEEE Trans. on PAMI*, 24(3):301–312, 2002.

[17] D. Modha and W. Scott-Spangler. Feature weighting in k-means clustering. *Machine Learning*, 2002. to appear.

[18] S. Roberts, C. Holmes, and D. Denison. Minimum-entropy data partitioning using RJ-MCMC. *IEEE Trans. on PAMI*, 23(8):909-914, 2001.

[19] L. Talavera. Dependency-based feature selection for clustering symbolic data. *Intelligent Data Analysis*, 4:19–28, 2000.

[20] G. Trunk. A problem of dimensionality: A simple example. *IEEE Trans. on PAMI*, 1(3):306–307, 1979.

[21] S. Vaithyanathan and B. Dom. Generalized model selection for unsupervised learning in high dimensions. In S. Solla, T. Leen, and K. Muller, eds, *Proc. of NIPS'12*. MIT Press, 2000.

[22] E. Xing, M. Jordan, and R. Karp. Feature selection for high-dimensional genomic microarray data. In *Proc. ICML'2001*, pp. 601–608, 2001.

[23] C. Wallace and P. Freeman. Estimation and inference via compact coding. *Journal of the Royal Statistical Society (B)*, 49(3):241–252, 1987.

[24] C.S. Wallace and D.L. Dowe. MML clustering of multi-state, Poisson, von Mises circular and Gaussian distributions. *Statistics and Computing*, 10:73–83, 2000.
